# GENERALIZATION OF BACKPROPAGATION
# TO
# RECURRENT AND HIGHER ORDER NEURAL NETWORKS

Fernando J. Pineda

*Applied Physics Laboratory, Johns Hopkins University*
*Johns Hopkins Rd., Laurel MD 20707*

## Abstract

A general method for deriving backpropagation algorithms for networks with recurrent and higher order networks is introduced. The propagation of activation in these networks is determined by dissipative differential equations. The error signal is backpropagated by integrating an associated differential equation. The method is introduced by applying it to the recurrent generalization of the feedforward backpropagation network. The method is extended to the case of higher order networks and to a constrained dynamical system for training a content addressable memory. The essential feature of the adaptive algorithms is that adaptive equation has a simple outer product form.

Preliminary experiments suggest that learning can occur very rapidly in networks with recurrent connections. The continuous formalism makes the new approach more suitable for implementation in VLSI.

## Introduction

One interesting class of neural networks, typified by the Hopfield neural networks [1,2] or the networks studied by Amari[3,4] are dynamical systems with three salient properties. First, they posses very many degrees of freedom, second their dynamics are nonlinear and third, their dynamics are dissipative. Systems with these properties can have complicated attractor structures and can exhibit computational abilities.

The identification of attractors with computational objects, e.g. memories ar d rules, is one of the foundations of the neural network paradigm. In this paradigm, programming becomes an excercise in manipulating attractors. A learning algorithm is a rule or dynamical equation which changes the locations of fixed points to encode information. One way of doing this is to minimize, by gradient descent, some function of the system parameters. This general approach is reviewed by Amari[4] and forms the basis of many learning algorithms. The formalism described here is a specific case of this general approach.

The purpose of this paper is to introduce a formalism for obtaining adaptive dynamical systems which are based on backpropagation[5,6,7]. These dynamical systems are expressed as systems of coupled first order differential equations. The formalism will be illustrated by deriving adaptive equations for a recurrent network with first order neurons, a recurrent network with higher order neurons and finally a recurrent first order associative memory.

## Example 1: Recurrent backpropagation with first order units

Consider a dynamical system whose state vector **x** evolves according to the following set of coupled differential equations

$$dx_i/dt = -x_i + g_i(\sum_j w_{ij}x_j) + I_i \qquad (1)$$

where i=1,...,N. The functions $g_i$ are assumed to be differentiable and may have different forms for various populations of neurons. In this paper we shall make no other requirements on $g_i$. In the neural network literature it is common to take these functions to be sigmoid shaped functions. A commonly used form is the logistic function,

$$g(\xi) = (1 + e^{-\xi})^{-1}. \qquad (2)$$

This form is biologically motivated since it attempts to account for the refractory phase of real neurons. However, it is important to stress that there is nothing in the mathematical content of this paper which requires this form -- any differentiable function will suffice in the formalism presented in this paper. For example, a choice which may be of use in signal processing is $\sin(\xi)$.

A necessary condition for the learning algorithms discussed here to exist is that the system posesses stable isolated attractors, i.e. fixed points. The attractor structure of (1) is the same as the more commonly used equation

$$du_i/dt = -u_i + \sum_j w_{ij}g(u_j) + K_i. \qquad (3)$$

Because (1) and (3) are related by a simple linear transformation. Therefore results concerning the stability of (3) are applicable to (1). Amari[3] studied the dynamics of equation (3) in networks with random conections. He found that collective variables corresponding to the mean activation and its second moment must exhibit either stable or bistable behaviour. More recently, Hopfield[2] has shown how to construct content addressable memories from symmetrically connected networks with this same dynamical equation. The symmetric connections in the network gaurantee global stability. The solution of equation (1) is also globally asymptotically stable if **w** can be transformed into a lower triangular matrix by row and column exchange operations. This is because in such a case the network is a simply a feedforward network and the output can be expressed as an explicit function of the input. No Liapunov function exists for arbitrary weights as can be demonstrated by constructing a set of weights which leads to oscillation. In practice, it is found that oscillations are not a problem and that the system converges to fixed points unless special weights are chosen. Therefore it shall be assumed, for the purposes of deriving the backpropagation equations, that the system ultimately settles down to a fixed point.

Consider a system of N neurons, or units, whose dynamics is determined by equation (1). Of all the units in the network we will arbitrarily define some subset of them (A) as *input* units and some other subset of them ($\Omega$) as *output* units. Units which are neither members of A nor $\Omega$ are denoted *hidden* units. A unit may be simultaneously an input unit and an output unit. The external environment influences the system through the source term, **I**. If a unit is an input unit, the corresponding component of **I** is nonzero. To make this more precise it is useful to introduce a notational convention. Suppose that $\Phi$ represent some subset of units in the network then the function $\Theta_{i\Phi}$ is defined by

$$\Theta_{i\Phi} = \begin{cases} 1 & \text{if i-th unit is a member of } \Phi \\ 0 & \text{otherwise} \end{cases} \qquad (4)$$

In terms of this function, the components of the **I** vector are given by

$$I_i = \xi_i\Theta_{iA} , \qquad (5)$$

where $\xi_i$ is determined by the external environment.

Our goal will be to find a local algorithm which adjusts the weight matrix **w** so that a given initial state $x^0 = x(t_0)$, and a given input **I** result in a fixed point, $x^\infty = x(t_\infty)$, whose components have a desired set of values $T_i$ along the output units . This will be accomplished by minimizing a function E which measures the distance between the desired fixed point and the actual fixed point i.e.,

$$E = \frac{1}{2} \sum_{i=1}^{N} J_i^2 \tag{6}$$

where

$$J_i = (T_i - x^\infty_i) \, \Theta_{i\Omega} \; . \tag{7}$$

E depends on the weight matrix **w** through the fixed point $x^\infty(w)$. A learning algorithm drives the fixed points towards the manifolds which satisfy $x_i^\infty = T_i$ on the output units. One way of accomplishing this with dynamics is to let the system evolve in the weight space along trajectories which are antiparallel to the gradient of E. In other words,

$$dw_{ij}/dt = - \eta \frac{\partial E}{\partial w_{ij}} \tag{8}$$

where $\eta$ is a numerical constant which defines the (slow) time scale on which **w** changes. $\eta$ *must be small so that* **x** *is always essentially at steady state* , i.e. $x(t) \cong x^\infty$. It is important to stress that the choice of gradient descent for the learning dynamics is by no means unique, nor is it necessarily the best choice. Other learning dynamics which employ second order time derivatives (e.g. the momentum method[5]) or which employ second order space derivatives (e.g. second order backpropagation[8]) may be more useful in particular applications. However, equation (8) does have the virtue of being the simplest dynamics which minimizes E.

On performing the differentiations in equation (8), one immediately obtains

$$dw_{rs}/dt = \eta \sum_k J_k \frac{\partial x^\infty_k}{\partial w_{rs}} \; . \tag{9}$$

The derivative of $x^\infty_k$ with respect to $w_{rs}$ is obtained by first noting that the fixed points of equation (1) satisfy the nonlinear algebraic equation

$$x^\infty_i = g_i(\sum_j w_{ij} x^\infty_j) + I_i \; , \tag{10}$$

differentiating both sides of this equation with respect to $w_{rs}$ and finally solving for $\partial x^\infty_k / \partial w_{rs}$. The result is

$$\frac{\partial x^\infty_k}{\partial w_{rs}} = (L^{-1})_{kr} \, g_r'(u_r) x^\infty_s \tag{11}$$

where $g_r'$ is the derivative of $g_r$ and where the matrix **L** is given by

$$L_{ij} = \delta_{ij} - g_i'(u_i) w_{ij} \; . \tag{12}$$

$\delta_{ij}$ is the Kroneker $\delta$ function ( $\delta_{ij} = 1$ if i=j, otherwise $\delta_{ij} = 0$). On substituting (11) into (9) one obtains the remarkably simple form

$$dw_{rs}/dt = \eta \, y_r x^{\infty}{}_s \qquad (13)$$

where

$$y_r = g_r{}'(u_r) \, \Sigma J_k(L^{-1})_{kr} \quad . \qquad (14)$$
$$\phantom{y_r = g_r{}'(u_r)}_{k=}$$

Equations (13) and (14) specify a formal learning rule. Unfortunately, equation (14) requires a matrix inversion to calculate the error signals $y_k$. Direct matrix inversions are necessarily nonlocal calculations and therefore this learning algorithm is not suitable for implementation as a neural network. Fortunately, a local method for calculating $y_r$ can be obtained by the introduction of an associated dynamical system. To obtain this dynamical system first rewrite equation (14) as

$$\Sigma L_{rk}\{y_r / g_r{}'(u_r)\} = J_k \quad . \qquad (15)$$
$${}_r$$

Then multiply both sides by $f_k{}'(u_k)$, substitute the explicit form for $L$ and finally sum over r. The result is

$$0 = -y_k + g_k{}'(u_k)\{\Sigma w_{rk} y_r + J_k\} \quad . \qquad (16)$$
$$\phantom{0 = -y_k + g_k{}'(u_k)\{}_r$$

One now makes the observation that the solutions of this linear equation are the fixed points of the dynamical system given by

$$dy_k/dt = -y_k + g_k{}'(u_k)\{\Sigma w_{rk} y_r + J_k\} \quad . \qquad (17)$$
$$\phantom{dy_k/dt = -y_k + g_k{}'(u_k)\{}_r$$

This last step is not unique, equation (16) could be transformed in various ways leading to related differential equations, cf. Pineda[9]. It is not difficult to show that the first order finite difference approximation (with a time step $\Delta t = 1$) of equations (1), (13) and (17) has the same form as the conventional backpropagation algorithm.

Equations (1), (13) and (17) completely specify the dynamics for an adaptive neural network, provided that (1) and (17) converge to stable fixed points and provided that both quantities on the right hand side of equation (13) are the steady state solutions of (1) and (17).

It was pointed out by Almeida[10] that the local stability of (1) is a sufficient condition for the local stability of (17). To prove this it suffices to linearize equation (1) about a stable fixed point. The resulting linearized equation depends on the same matrix $L$ whose transpose appears in the derivation of equation (17), cf. equation (15). But $L$ and $L^T$ have the same eigenvalues, hence it follows that the fixed points of (17) must also be locally stable if the fixed points of (1) are locally stable.

## Learning multiple associations

It is important to stress that up to this point the entire discussion has assumed that $I$ and $T$ are constant in time, thus no mechanism has been obtained for learning multiple input/output associations. Two methods for training the network to learn multiple associations are now discussed. These methods lead to qualitatively different learning behaviour.

Suppose that each input/output pair is labeled by a pattern label $\alpha$, i.e. $\{I^\alpha, T^\alpha\}$. Then the energy function which is minimized in the above discussion must also depend on this label since it is an implicit function of the $I^\alpha, T^\alpha$ pairs. In order to learn multiple input/output associations it is necessary to minimize all the $E[\alpha]$ simultaniously. In otherwords the function to minimize is

$$E_{total} = \Sigma \, E[\alpha] \qquad (18)$$
$$\phantom{E_{total} =}_{\alpha}$$

where the sum is over all input/output associations. From (18) it follows that the gradient for $E_{total}$ is simply the sum of the gradients for each association, hence the corresponding gradient descent equation has the form,

$$dw_{ij}/dt = \eta \sum_{\alpha} y^{\infty}_{i}[\alpha] \, x^{\infty}_{j}[\alpha] \, . \qquad (19)$$

In numerical simulations, each time step of (19) requires relaxing (1) and (17) for each pattern and accumulating the gradient over all the patterns. This form of the algorithm is deterministic and is guaranteed to converge because, by construction, $E_{total}$ is a Liapunov function for equation (19). However, the system may get stuck in a local minimum. This method is similar to the master/slave approach of Lapedes and Farber[11]. Their adaptive equation, which plays the same role as equation (19), also has a gradient form, although it is not strictly descent along the gradient. For a randomly or fully connected network it can be shown that the number of operations required per weight update in the master/slave formalism is proportional to $N^3$ where N is the number of units. This is because there are $O(N^2)$ update equations and each equation requires $O(N)$ operations (assuming some precomputation). On the other hand, in the backpropagation formalism each update equation requires only $O(1)$ operations because of their trivial outer product form. Also $O(N^2)$ operations are required to precompute $x^{\infty}$ and $y^{\infty}$. The result is that each weight update requires only $O(N^2)$ operations. It is not possible to conclude from this argument that one or the other approach will be more efficient in a particular application because there are other factors to consider such as the number of patterns and the number of time steps required for x and y to converge. A detailed comparison of the two methods is in preparation.

A second approach to learning multiple patterns is to use (13) and to change the patterns *randomly* on each time step. The system therefore receives a sequence of random impulses each of which attempts to minimize $E[\alpha]$ for a single pattern. One can then define L(w) to be the mean $E[\alpha]$ (averaged over the distribution of patterns).

$$L(w) = <E \, [w, I^{\alpha}, T^{\alpha}]> \qquad (20)$$

Amari[4] has pointed out that if the sequence of random patterns is stationary and if L(w) has a unique minimum then the theory of stochastic approximation guarantees that the solution of (13) w(t) will converge to the minimum point $w_{min}$ of L(w) to within a small fluctuating term which vanishes as $\eta$ tends to zero. Evidently $\eta$ is analogous to the temperature parameter in simulated annealing. This second approach generally converges more slowly than the first, but it will ultimately converge (in a statistical sense) to the *global* minimum.

In principle the fixed points, to which the solutions of (1) and (17) eventually converge, depend on the initial states. Indeed, Amari's[3] results imply that equation (1) is bistable for certain choices of weights. Therefore the presentation of multiple patterns might seem problematical since in both approaches the final state of the previous pattern becomes the initial state of the new pattern. The safest approach is to reinitialize the network to the same initial state each time a new pattern is presented, e.g. $x_i(t_o) = 0.5$ for all i. In practice the system learns robustly even if the initial conditions are chosen randomly.

**Example 2: Recurrent higher order networks**

It is straightforward to apply the technique of the previous section to a dynamical system with higher order units. Higher order systems have been studied by Sejnowski [12] and Lee et al.[13]. Higher order networks may have definite advantages

over networks with first order units alone  A detailed discussion of the backpropagation formalism applied to higher order networks is beyond the scope of this paper.  Instead, the adaptive equations for a network with purely n-th order units will be presented as an example of the formalism.  To this end consider a dynamical system of the form

$$dx_i/dt = -x_i + g_i(u_i) + I_i \tag{21}$$

where

$$u_i = \sum_j \cdots \sum_k w^{(n)}_{ij\cdots k} f_j(x_j) \cdots f_k(x_k) \ . \tag{22}$$

and where there are n+1 indices and the summations are over all indices except i. The superscript on the weight tensor indicates the order of the correlation. Note that an additional nonlinear function f has been added to illustrate a further generalization. Both f and g must be differentiable and may be chosen to be sigmoids. It is not difficult, although somewhat tedious, to repeat the steps of the previous example to derive the adaptive equations for this system. The objective function in this case is the same as was used in the first example, i.e. equation (6). The n-th order gradient descent equation has the form

$$dw^{(n)}_{ij\cdots k}/dt = \eta y^{(n)\infty}_i f(x^\infty_j) \cdots f(x^\infty_k) \ . \tag{23}$$

Equation (23) illustrates the major feature of backpropagation which distinguishes it from other gradient descent algorithms or similar algorithms which make use of a gradient.  Namely, that the gradient of the objective function has a very trivial outer product form.  $y^{(n)\infty}$ is the steady state solution of

$$dy^{(n)}_k/dt = - y^{(n)}_k + g_k'(u_k)\{f_k'(x_k)\sum_r V^{(n)}_{rk} y^{(n)}_r + J_k\} \ . \tag{24}$$

The matrix $V^{(n)}$ plays the role of $w$ in the previous example, however $V^{(n)}$ now depends on the state of the network according to

$$V^{(n)}_{ij} = \sum_k \cdots \sum_l S^{(n)}_{ijk\cdots l} \{ f(x_k) \cdots f(x_l)\} \tag{25}$$

where is $S^{(n)}$ a tensor which is symmetric with respect to the exchange of the second index and all the indices to the right, i.e.

$$S^{(n)}_{ijk\cdots l} = w^{(n)}_{ijk\cdots l} + w^{(n)}_{ikj\cdots l} + \cdots + w^{(n)}_{ijl\cdots k} \ . \tag{26}$$

Finally, it should be noted that: 1) If the polynomial $u_i$ is not homogenous, the adaptive equations are more complicated and involve cross terms between the various orders and that: 2) The local stability of the n-th order backpropagation equations now depends on the eigenvalues of the matrix

$$L_{ij} = \delta_{ij} - g_i'(u_i) f_i'(x_i) V^{(n)}_{ij} \ . \tag{27}$$

As before, if the forward propagation converges so will the backward propagation.

**Example 3: Adaptive content addressable memory**

In this section the adaptive equations for a content addressable memory (CAM) are derived as a final illustration of the generality of the formalism. Perhaps

the best known (and best studied) examples of dynamical systems which exhibit CAM behaviour are the systems discussed by Hopfield[1,2]. Hopfield used a nonadaptive method for programming the symmetric weight matrix. More recently Lapedes and Farber[11] have demonstrated how to contruct a master dynamical system which can be used to train the weights of a slave system which has the Hopfield form. This slave system then performs the CAM operation. The resulting weights are not symmetric.

The learning proceedure presented in this section is most closely related to the method of Lapedes and Farber in that a master network is used to adjust the weights of a slave network. In constrast to the afforementioned formalism, which requires a very large associated weight matrix for the master network, both the master and slave networks of the following approach make use of the same weight matrix. The CAM under consideration is based on equation (1). However, the interpretation of the dynamics will be somewhat different from the first section. The main difference is that the dynamics in the learning phase is constrained. The constrained dynamical system is denoted the master network. The unconstrained system is denoted the slave network. The units in the network are divided into only two sets: the set of visible units (V) and the set of internal or hidden units (H). There will be no distinction made between input and output units. Thus, **I** will generally be zero unless an input bias is needed in some application.

The dynamical system will be used as an autoassociative memory, thus the memory recall is performed by starting the network at a particular initial state which represents partial information about a stored memory. More precisely, suppose that there exists a subset K of the visible units whose states are known to have values $T_i$. Then the initial state of the network is

$$x_i(t_o) = T_i \, \Theta_{iK} + b_i \, (1 - \Theta_{iK}), \tag{28}$$

where the $b_i$ are arbitrary. The CAM relaxes to the previously stored memory whose basin of attraction contains this partial state.

Memories are stored by a master network whose topology is exactly the same as the slave network, but whose dynamics is somewhat modified. The state vector **z** of the master network evolves according to the equation

$$dz_i/dt = -z_i + g_i(\sum_{k=1}^{N} w_{ik} Z_k) + I_i \tag{29}$$

where **Z** is defined by

$$Z_i = T_i \, \Theta_{iV} + z_i \, \Theta_{iH} . \tag{30}$$

The components of **Z** along the visible units are just the target value specified by **T**. This equation is useful as a master equation because if the weights can be chosen so that the $z_i$ of the visible units relax to the target values $T_i$, then a fixed point of (29) is also a fixed point of (1). It can be concluded therefore, that by training the weights of the master network one is also training the weights of the slave network. Note that the form of **Z** implies that equation (29) can be rewritten as

$$dz_i/dt = -z_i + g_i(\sum_{k \in H} w_{ik} z_k - \theta_i) + I_i \tag{31}$$

where

$$\theta_i = -\sum_{k \in V} w_{ik} T_k . \tag{32}$$

From equations (31) and (32) it is clear that the dynamics of the master system is driven by the thresholds which depend on the targets.

To derive the adaptive equations consider the objective function

$$E_{master} = \frac{1}{2} \sum_{i=1}^{N} J_i^2 .$$ (33)

where

$$J_i = Z^{\infty}_i - z^{\infty}_i .$$ (34)

It is straightforward to apply the steps discussed in previous sections to $E_{master}$. This results in adaptive equations for the weights. The mathematical details will be omitted since they are essentially the same as before, the gradient descent equation is

$$dw_{ij}/dt = \eta y^{\infty}_i Z^{\infty}_j$$ (35)

where $y^{\infty}$ is the steady state solution of

$$dy_k/dt = - y_k + g'_k(v_k)\{\Theta_{iH}\sum_r w_{rk}y_r + J_k\}$$ (36)

where

$$v_i = \sum_{i \in H} w_{ik}Z^{\infty}_k .$$ (37)

Equations (31), and (35)-37) define the dynamics of the master network. To train the slave network to be an autoassociative memory it is necessary to use the stored memories as the initial states of the master network, i.e.

$$z_i(t_o) = T_i \Theta_{iV} + b_i \Theta_{iH}$$ (39)

where $b_i$ is an arbitrary value as before. The previous discussions concerning the stability of the three equations (1), (13) and (17) apply to equations (31) (35) and (36) as well. It is also possible to derive the adaptive equations for a higher order associative network, but this will not be done here.

Only preliminary computer simulations have been performed with this algorithm to verify their validity, but more extensive experiments are in progress. The first simulation was with a fully connected network with 10 visible units and 5 hidden units. The training set consisted of four random binary vectors with the magnitudes of the vectors adjusted so that $0.1 \leq T_i \leq 0.9$. The equations were approximated by first order finite difference equations with $\Delta t = 1$ and $\eta = 1$. The training was performed with the deterministic method for learning multiple associations. Figure 1. shows $E_{total}$ as a function of the number of updates for both the master and slave networks. $E_{total}$ for the slave exhibits discontinous behaviour because the trajectory through the weight space causes $x(t_o)$ to cut across the basins of attraction for the fixed points of equation (1).

The number of updates required for the network to learn the patterns is relatively modest and can be reduced further by increasing $\eta$. This suggests that learning can occur very rapidly in this type of network.

## Discussion

The algorithms presented here by no means exhaust the class of possible adaptive algorithms which can be obtained with this formalism. Nor is the choice of gradient descent a crucial feature in this formalism. The key idea is that it is possible to express the gradient of an objective function as the outer product of vectors which can be calculated by dynamical systems. This outer product form is also responsible for the fact that the gradient can be calculated with only $O(N^2)$ operations in a fully connected or randomly connected network. In fact the number of operations per weight update is proportional to the number of connections in the network. The methods used here will generalize to calculate higher order derivatives of the objective function as well.

The fact that the algorithms are expressed as differential equations suggests that they may be implemented in analog electronic or optical hardware.

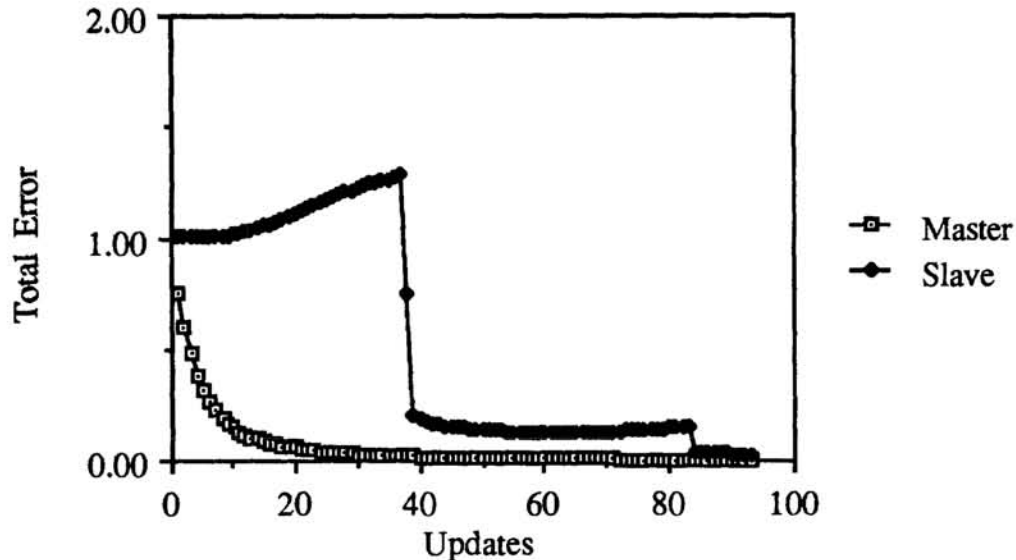

**figure 1.** $E_{total}$ as a function of the the number of updates.

## References

(1)    J. J. Hopfield, *Neural Networks as Physical Systems with Emergent Collective Computational Abilities*, Proc. Nat. Acad. Sci. USA, Bio.79, 2554-2558, (1982)

(2)    J. J. Hopfield, *Neurons with graded response have collective computational properties like those of two-state neurons*, Proc. Nat. Acad. Sci. USA, Bio. 81, 3088-3092, (1984)

(3)    Shun-Ichi Amari, IEEE Trans. on Systems Man and Cybernetics, 2, 643-657, (1972)

(4)    Shun-Ichi Amari, in Systems Neuroscience, ed. Jacqueline Metzler, Academic press, (1977)

(5)    D. E. Rumelhart, G. E. Hinton and R.J. Williams, in Parallel Distributed Processing, edited by D. E. Rumelhart and J. L. McClelland, M.I.T. press, (1986)

(6)    David B. Parker, *Learning-Logic*, Invention Report, S81-64, File 1, Office of Technology Licensing, Stanford University, October, 1982

(7)    Y. LeChun, Proceedings of Cognitiva, 85, p. 599, (1985)

(8)    David B. Parker, *Second Order Backpropagation: Implementing an Optimal O(n) Approximation to Newton's Method as an Artificial Neural Network*, submitted to Computer, (1987)

(9)    Fernando J. Pineda, *Generalization of backpropagation to recurrent neural networks*, Phys. Rev. Lett., 18, 2229-2232, (1987)

(10)   Luis B. Almeida, in the Proceedings of the IEEE First Annual International Conference on Neural Networks, San Diego, California, June 1987, edited by

M. Caudil and C. Butler (to be published This is a discrete version of the algorithm presented as the first example

(11) Alan Lapedes and Robert Farber, *A self-optimizing, nonsymmetrical neural net for content addressable memory and pattern recognition*, Physica, D22, 247-259, (1986), see also, *Programming a Massively Parallel, Computation Universal System: Static Behaviour*, in Neural Networks for Computing Snowbird, UT 1986, AIP Conference Proceedings, 151, (1986), edited by John S. Denker

(12) Terrence J. Sejnowski, *Higher-order Boltzmann Machines*, Draft preprint obtained from author

(13) Y.C. Lee, Gary Doolen, H.H. Chen, G.Z. Sun, Tom Maxwell, H.Y. Lee and C. Lee Giles, *Machine Learning using a higher order correlation network*, Physica D22, 276-306, (1986)